# Direct memory access using two cues: Finding the intersection of sets in a connectionist model

Janet Wiles, Michael S. Humphreys, John D. Bain and Simon Dennis
Departments of Psychology and Computer Science
University of Queensland QLD 4072 Australia
email: janet@psych.psy.uq.oz.au

## Abstract

For lack of alternative models, search and decision processes have provided the dominant paradigm for human memory access using two or more cues, despite evidence against search as an access process (Humphreys, Wiles & Bain, 1990). We present an alternative process to search, based on calculating the intersection of sets of targets activated by two or more cues. Two methods of computing the intersection are presented, one using information about the possible targets, the other constraining the cue-target strengths in the memory matrix. Analysis using orthogonal vectors to represent the cues and targets demonstrates the competence of both processes, and simulations using sparse distributed representations demonstrate the performance of the latter process for tasks involving 2 and 3 cues.

## 1 INTRODUCTION

Consider a task in which a subject is asked to name a word that rhymes with *oast*. The subject answers "most", (or post, host, toast, boast, ...). Now the subject is asked to find a word that means *a mythical being that rhymes with oast*. She or he pauses slightly and replies "ghost".

The difference between the first and second questions is that the first requires the use of one cue to access memory. The second question requires the use of two cues – either combining them before the access process, or combining the targets they access. There are many experimental paradigms in psychology in which a subject uses two or more cues to perform a task (Rubin & Wallace, 1989). One default assumption underlying

many explanations for the effective use of two cues relies on a search process through memory.

Models of human memory based on associative access (using connectionist models) have provided an alternative paradigm to search processes for memory access using a single cue (Anderson, Silverstein, Ritz & Jones, 1977; McClelland & Rumelhart, 1986), and for two cues which have been studied together (Humphreys, Bain & Pike 1989). In some respects, properties of these models correspond very closely to the characteristics of human memory (Rumelhart, 1989). In addition to the evidence against search processes for memory access using a single cue, there is also experimental evidence against sequential search in some tasks requiring the combination of two cues, such as cued recall with an extra-list cue, cued recall with a part-word cue, lexical access and semantic access (Humphreys, Wiles & Bain, 1990). Furthermore, in some of these tasks it appears that the two cues have never jointly occurred with the target. In such a situation, the tensor product employed by Humphreys et. al. to bind the two cues to the target cannot be employed, nor can the co-occurrences of the two cues be encoded into the hidden layer of a three-layer network. In this paper we present the computational foundation for an alternative process to search and decision, based on parallel (or direct) access for the intersection of sets of targets that are retrieved in response to cues that have not been studied together.

*Definition of an intersection in the cue-target paradigm:* Given a set of cue-target pairs, and two (or more) access cues, then the intersection specified by the access cues is defined to be the set of targets which are associated with both cues. If the cue-target strengths are not binary, then they are constrained to lie between 0 and 1, and targets in the intersection are weighted by the product of the cue-target strengths. A complementary definition for a union process could be the set of targets associated with any one or more of the access cues, weighted by the sum of the target strengths.

In the models that are described below, we assume that the access cues and targets are represented as vectors, the cue-target associations are represented in a memory matrix and the set of targets retrieved in response to one or more cues is represented as a linear combination, or *blend*, of target vectors associated with that cue or cues. Note that under this definition, if there is more than one target in the intersection, then a second stage is required to select a unique target to output from the retrieved linear combination. We do not address this second stage in this paper.

*A task requiring intersection:* In the rhyming task described above, the rhyme and semantic cues have extremely low separate probabilities of accessing the target, *ghost*, but a very high joint probability. In this study we do not distinguish between the representation of the semantic and part-word cues, although it would be required for a more detailed model. Instead, we focus on the task of retrieving a target weakly associated with two cues. We simulate this condition in a simple task using two cues, $C_1$ and $C_2$, and three targets, $T_1$, $T_2$ and $T_3$. Each cue is strongly associated with one target, and weakly associated with a second target, as follows (strengths of association are shown above the arrows):

$$C_1 \xrightarrow{.9} T_1, \; C_1 \xrightarrow{.1} T_2, \; C_2 \xrightarrow{.1} T_2, \text{ and } C_2 \xrightarrow{.9} T_3.$$

The intersection of the targets retrieved to the two cues, $C_1$ and $C_2$, is the target, $T_2$, with a strength of 0.01. Note that in this example, a model based on vector addition would be insufficient to select target, $T_2$, which is weakly associated with both cues, in preference to either target, $T_1$ or $T_3$, which are strongly associated with one cue each.

# 2  IMPLEMENTATIONS OF INTERSECTION PROCESSES

## 2.1  LOCAL REPRESENTATIONS

Given a local representation for two sets of targets, their intersection can be computed by multiplying the activations elicited by each cue. This method extends to sparse representations with some noise from cross product terms, and has been used by Dolan and Dyer (1989) in their tensor model, and Touretzky and Hinton (1989) in the Distributed Connectionist Production System (for further discussion see Wiles, Humphreys, Bain & Dennis, 1990). However, multiplying activation strengths does not extend to fully distributed representations, since multiplication depends on the basis representation (i.e., the target patterns themselves) and the cross-product terms do not necessarily cancel. One strong implication of this for implementing an intersection process, is that the choice of patterns is not critical in a linear process (such as vector addition) but can be critical in a non-linear process (which is necessary for computing intersections). An intersection process requires more information about the target patterns themselves.

It is interesting to note that the inner product of the target sets (equivalent to the match process in Humphreys et. al.'s (1989) Matrix model) can be used to determine whether or not the intersection of targets is empty, if the target vectors are orthogonal, although it cannot be used to find the particular vectors which are in the intersection.

## 2.2  USING INFORMATION ABOUT TARGET VECTORS

A local representation enables multiplication of activation strengths because there is implicit knowledge about the allowable target vectors in the local representation itself. The first method we describe for computing the intersection of fully distributed vectors uses information about the targets, explicitly represented in an auto-associative memory, to filter out cross-product terms: In separate operations, each cue is used to access the memory matrix and retrieve a composite target vector (the linear combination of associated targets). A temporary matrix is formed from the outer product of these two composite vectors. This matrix will contain product terms between all the targets in the intersection set as well as noise in the form of cross-product terms. The cross-product terms can be filtered from the temporary matrix by using it as a retrieval cue for accessing a three-dimensional auto-associator (a tensor of rank 3) over all the targets in the original memory. If the target vectors are orthonormal, then this process will produce a vector which contains no noise from cross-product terms, and is the linear combination of all targets associated with both cues (see Box 1).

*Box 1. Creating a temporary matrix from the product of the target vectors, then filtering out the noise terms*: Let the cues and targets be represented by vectors which are mutually orthonormal (i.e., $C_i.C_i = T_i.T_i = 1$, $C_i.C_j = T_i.T_j = 0$, $i, j = 1,2,3$). The memory matrix, $\mathbf{M}$, is formed from cue-target pairs, weighted by their respective strengths, as follows:

$$\mathbf{M} = 0.9C_1 T_1' + 0.1C_1 T_2' + 0.1C_2 T_2' + 0.9C_2 T_3'$$

where $T'$ represents the transpose of $T$, and $C_i T_i'$ is the outer product of $C_i$ and $T_i$.
In addition, let $\mathbf{Z}$ be a three-dimensional auto-associative memory (or tensor of rank 3) created over three orthogonal representations of each target (i.e., $T_i$ is a column vector, $T_i'$ is a row vector which is the transpose of $T_i$, and $T_i''$ is the vector in a third direction orthogonal to both, where $i=1,2,3$), as follows:

$$\mathbf{Z} = \Sigma_i T_i T_i' T_i''$$

Let a two-dimensional temporary matrix, $\mathbf{X}$, be formed by taking the outer product of target vectors retrieved to the access cues, as follows:

$$\mathbf{X} = (C_1 \mathbf{M})(C_2 \mathbf{M})'$$
$$= (0.9T_1 + 0.1T_2)(0.1T_2 + 0.9T_3)'$$
$$= 0.09T_1T_2' + 0.81T_1T_3' + 0.01T_2T_2' + 0.09T_2T_3'$$

Using the matrix $\mathbf{X}$ to access the auto-associator $\mathbf{Z}$, will produce a vector from which all the cross-product terms have been filtered, as follows:

$$\mathbf{X}\,\mathbf{Z} = (0.09T_1T_2' + 0.81T_1T_3' + 0.01T_2T_2' + 0.09T_2T_3')\ (\Sigma_i T_i T_i' T_i'')$$
$$= (0.09T_1T_2')(\Sigma_i T_i T_i' T_i'') + (0.81T_1T_3')(\Sigma_i T_i T_i' T_i'')$$
$$+ (0.01T_2T_2')(\Sigma_i T_i T_i' T_i'') + (0.09T_2T_3')(\Sigma_i T_i T_i' T_i'')$$
$$= (0.01T_2T_2')(T_2 T_2' T_2'') \qquad \text{since all other terms cancel.}$$
$$= 0.01T_2''$$

This vector is the required intersection of the linear combination of target vectors associated with both the input cues, $C_1$ and $C_2$ weighted by the product of the strengths of associations from the cues to the targets.

A major advantage of the above process is that only matrix (or tensor) operations are used, which simplifies both the implementation and the analysis. The behaviour of the system can be analysed either at the level of behaviours of patterns, or using a coordinate system based on individual units, since in a linear system these two levels of description are isomorphic. In addition, the auto-associative target matrix could be created incrementally when the target vectors are first learnt by the system using the matrix memory. The

disadvantages include the requirement for dynamic creation and short term storage of the two dimensional product-of-targets matrix, and the formation and much longer term storage of the three dimensional auto-associative matrix. It is possible, however, that an auto-associator may be part of the output process.

## 2.3 ADDITIVE APPROXIMATIONS TO MULTIPLICATIVE PROCESSES

An alternative approach to using the target auto-associator for computing the intersection, is to incorporate a non-linearity at the time of memory storage, rather than memory access. The aim of this transform would be to change the cue-target strengths so that linear addition of vectors could be used for computing the intersection. An operation that is equivalent to multiplication is the addition of logarithms. If the logarithm of each cue-target strength was calculated and stored at the time of association, then an additive access process would retrieve the intersection of the inputs. More generally, it may be possible to use an operation that preserves the same order relations (in terms of strengths) as multiplication. It is always possible to find a restricted range of association strengths such that the sum of a number of weak cue-target associations will produce a stronger target activation than the sum of a smaller number of strong cue-target associations. For example, by scaling the target strengths to the range $[(n-1)/n, 1]$ where $n$ is the number of simultaneously available cues, vector addition can be made to approximate multiplication of target strengths.

This method has the advantage of extending naturally to non-orthogonal vectors, and to the combination of three or more cues, with performance limits determined solely by cross-talk between the vectors. Time taken is proportional to the number of cues, and noise is proportional to the product of the set sizes and cross-correlation between the vectors.

# 3 SIMULATIONS OF THE ADDITIVE PROCESS

Two simulations of the additive process using scaled target strengths were performed to demonstrate the feasibility of the method producing a target weakly associated with two cues, in preference to targets with much higher probabilities of being produced in response to a single cue. As a work-around for the problem of how (and when) to decompose the composite output vector, the target with the strongest correlation with the composite output was selected as the winner. To simulate the addition of some noise, non-orthogonal vectors were used.

The first simulation involved two cues, $C_1$ and $C_2$, and three targets, $T_1$, $T_2$ and $T_3$, represented as randomly generated 100 dimensional vectors, 20% 1s, the remainder 0s. Cue $C_1$ was strongly associated with target $T_1$ and weakly associated with target $T_2$, cue $C_2$ was strongly associated with target $T_3$ and weakly associated with target $T_2$. A trial consisted of generating random cue and target vectors, forming a memory matrix from their outer products (multiplied by 0.9 for strong associates and 0.6 for weak associates; note that these strengths have been scaled to the range, [0,1]), and then pre-multiplying the memory matrix by the appropriate cue (i.e., either $C_1$ or $C_2$ or $C_1 + C_2$).

The memory matrix, **M**, was formed as shown in Box 1. Retrieval to a cue, $C_1$, was as follows: $C_1 \mathbf{M} = 0.9\, C_1.C_1 T_1' + 0.6\, C_1.C_1 T_2' + 0.6\, C_1.C_2 T_2' + 0.9\, C_1.C_2 T_3'$. In this case, the cross product terms, $C_1.C_2$, do not cancel since the vectors are not orthogonal, although their expected contribution to the output is small (expected correlation 0.04). The winning target vector was the one that had the strongest correlation (smallest normalized dot product) with the resulting output vector. The results are shown in Table 1.

Table 1: Number of times each target was retrieved in 100 trials.

|        | t1 | t2 | t3 |
|--------|----|----|----|
| c1     | 92 | 8  | 0  |
| c2     | 0  | 9  | 91 |
| c1+c2  | 11 | 80 | 9  |

Over 100 trials, the results show that when either cue $C_1$ or $C_2$ was presented alone, the target with which it was most strongly paired was retrieved in over 90% of cases. Target $T_2$ had very low probabilities of recall given either $C_1$ or $C_2$ (8% and 9% respectively), however, it was very likely to be recalled if both cues were presented (80%).

The first simulation demonstrated the multi-cue paradigm with the simple two-cue and three-target case. In a second simulation, the system was tested for robustness in a similar case involving three cues, $C_1$ to $C_3$, and four targets, $T_1$ to $T_4$. The results show that $T_4$ had low probabilities of recall given either $C_1$, $C_2$ or $C_3$ (13%, 22% and 18% respectively), medium probabilities of recall given a combination of two cues (36%, 31% and 28%), and was most likely to be recalled if all three cues were presented (44%). For this task, when three cues are presented concurrently, in the ideal intersection only $T_4$ should be produced. The results show that it is produced more often than the other targets (44% compared with 22%, 18% and 16%), each of which is strongly associated with two out of the three cues, but there is considerably more noise than in the two-cue case. (See Wiles, Humphreys, Bain & Dennis, 1990, for further details.)

## 4 DISCUSSION

The simulation results demonstrated the effect of the initial scaling of the cue-target strengths, and non-linear competition between the target outputs. It is important to note the difference between the *association strengths* from cues to targets and the *cued recall probability* of each target. In memory research, the association strengths have been traditionally identified with the probability of recall. However, in a connectionist model the association strengths are related to the weights in the network and the cued recall probability is the probability of recall of a given target to a given cue.

This paper builds on the idea that direct access is the default access method for human memory, and that all access processes are cue based. The immediate response from memory is a blend of patterns, which provide a useful intermediate stage. Other processes may act on the blend of patterns before a single target is selected for output in a

successive stage. One such process that may act on the intermediate representation is an intersection process that operates over blends of targets. Such a process would provide an alternative to search as a computational technique in psychological paradigms that use two or more cues. We don't claim that we have described the way to implement such a process – much more is required to investigate these issues. The two methods presented here have served to demonstrate that direct access intersection is a viable neural network technique. This demonstration means that more processing can be performed in the network dynamics, rather than by the control structures that surround memory.

## Acknowledgements

Our thanks to Anthony Bloesch, Michael Jordan, Julie Stewart, Michael Strasser and Roland Sussex for discussions and comments. This work was supported by grants from the Australian Research Council, a National Research Fellowship to J. Wiles and an Australian Postgraduate Research Award to S. Dennis.

## References

Anderson, J.A., Silverstein, J.W., Ritz, S.A. and Jones, R.S. Distinctive features, categorical perception, and probability learning: Some applications of a neural model. *Psychological Review, 84*, 413-451, 1977.

Dolan, C. and Dyer, M.G. Parallel retrieval and application of conceptual knowledge. *Proceedings of the 1988 Connectionist Models Summer School, San Mateo, Ca: Morgan Kaufmann*, 273-280, 1989.

Humphreys, M.S., Bain, J.D. and Pike, R. Different ways to cue a coherent memory system: A theory for episodic, semantic and procedural tasks. *Psychological Review, 96:2*, 208-233, 1989.

Humphreys, M.S., Wiles, J. and Bain, J.D. Direct Access: Cues with separate histories. Paper presented at Attention and Performance 14, Ann Arbor, Michigan, July, 1990.

McClelland, J.L. and Rumelhart, D.E. A distributed model of memory. In McClelland, J.L. and Rumelhart, D.E. (eds.) *Parallel Distributed Processing: Explorations in the microstructure of cognition*, 170-215, MIT Press, Cambridge, MA, 1986.

Rubin, D.C. and Wallace, W.T. Rhyme and reason: Analysis of dual retrieval cues. *Journal of Experimental Psychology: Learning, Memory and Cognition, 15:4*, 698-709, 1989.

Rumelhart, D.S. The architecture of mind: A connectionist approach. In Posner, M.I. (ed.) *Foundations of Cognitive Science*, 133-159, MIT Press, Cambridge, MA, 1989.

Touretzky, D.S. and Hinton, G.E. A distributed connectionist production system. *Cognitive Science, 12*, 423-466, 1988.

Wiles, J., Humphreys, M.S., Bain, J.D. and Dennis, S. Control processes and cue combinations in a connectionist model of human memory. Department of Computer Science Technical Report, #186, University of Queensland, October 1990, 40pp.